# FIXED POINT ANALYSIS FOR RECURRENT NETWORKS

Mary B. Ottaway

Patrice Y. Simard
Dept. of Computer Science
University of Rochester
Rochester NY 14627

Dana H. Ballard

## ABSTRACT

This paper provides a systematic analysis of the recurrent backpropagation (RBP) algorithm, introducing a number of new results. The main limitation of the RBP algorithm is that it assumes the convergence of the network to a stable fixed point in order to backpropagate the error signals. We show by experiment and eigenvalue analysis that this condition can be violated and that chaotic behavior can be avoided. Next we examine the advantages of RBP over the standard backpropagation algorithm. RBP is shown to build stable fixed points corresponding to the input patterns. This makes it an appropriate tool for content addressable memories, one-to-many function learning, and inverse problems.

## INTRODUCTION

In the last few years there has been a great resurgence of interest in neural network learning algorithms. One of the most successful of these is the Backpropagation learning algorithm of [Rumelhart 86], which has shown its usefulness in a number of applications. This algorithm is representative of others that exploit internal units to represent very nonlinear decision surfaces [Lippman 87] and thus overcomes the limits of the classical perceptron [Rosenblatt 62].

With its enormous advantages, the backpropagation algorithm has a number of disadvantages. Two of these are the inability to fill in patterns and the inability to solve one-to-many inverse problems [Jordan 88]. These limitations follow from the fact that the algorithm is only defined for a feedforward network. Thus if part of the pattern is missing or corrupted in the input, this error will be propagated through to the output and the original pattern will not be restored. In one-to-many problems, several solutions are possible for a given input. On a feedforward net, the competing targets for a given input introduce contradictory error signals and learning in unsuccessful.

Very recently, these limitations have been removed with the specification of a recurrent backpropagation algorithm [Pineda 87]. This algorithm effectively extends the backpropagation idea to networks of arbitrary connection topologies. This advantage, however, does not come without some risk. Since the connections in the network are not symmetric, the stability of the network is not guaranteed. For some choices of weights, the state of the units may oscillate indefinitely.

This paper provides a systematic analysis of the recurrent backpropagation (RBP) algorithm, introducing a number of new results. The main limitation of the RBP algorithm is that it assumes the convergence of the network to a stable fixed point in order to

backpropagate the error signals. We show by experiment and eigenvalue analysis that this condition can be violated and that chaotic behavior can be avoided.

Next we examine the advantage in convergence speed of RBP over the standard backpropagation algorithm. RBP is shown to build stable fixed points corresponding to the input patterns. This makes it an appropriate tool for content addressable memories, many-to-one function learning and inverse problem.

## MODEL DESCRIPTION

The simulations have been done on a recurrent backpropagation network with first order units. Using the same formalism as [Pineda 87], the vector state $x$ is updated according to the equation:

$$\mathrm{d}x_i/\mathrm{d}t = -x_i + g_i(u_i) + I_i \tag{1}$$

where

$$u_i = \sum_j w_{ij} x_j \quad \text{for} \quad i = 1, 2, \ldots, N \tag{2}$$

The activation function is the logistic function

$$g(\xi) = \frac{1}{1 + e^{-\xi}} \tag{3}$$

The networks we will consider are organized in modules (or sets) of units that perform similar functions. For example, we talk about fully connected module if each unit in the module is connected to each of the others. An input module is a set of units where each unit has non-zero input function $I_i$. Note that a single unit can belong to more than one module at a time. The performance of the network is measured through the energy function:

$$E = \frac{1}{2} \sum_{i=1}^{N} J_i^2 \tag{4}$$

where

$$J_i = (T_i - x_i^\infty) \quad \text{and} \quad x_i^\infty = x_i(t_\infty) \tag{5}$$

An output module is a set of units $i$ such that $J_i \neq 0$. Units that do not belong to any input or output modules are called hidden units. A unit (resp module) can be clamped and unclamped. When the unit (resp module) is unclamped, $I_i = J_i = 0$ for the unit (resp the module). If the unit is clamped, it behave according to the pattern presented to the network. Unclamping a unit results in making it hidden. Clamping and unclamping actions are handy concepts for the study of content addressable memory or generalization.

The goal for the network is to minimize the energy function by changing the weights accordingly. One way is to perform a gradient descent in $E$ using the delta rule:

$$\mathrm{d}w_{ij}/\mathrm{d}t = -\eta \frac{\partial E}{\partial w_{ij}} \tag{6}$$

where $\eta$ is a learning rate constant. The weight variation as a function of the error is given by the formula [Pineda 87, Almeida 87]

$$\mathrm{d}w_{ij}/\mathrm{d}t = \eta y_i^\infty x_j^\infty \tag{7}$$

where $y_i^\infty$ is a solution of the dynamical system

$$\mathrm{d}y_i/\mathrm{d}t = -y_i + g_i'(u_i^\infty)\left(\sum_k w_{ik}y_k + J_i\right) \tag{8}$$

The above discussion, assumes that the input function $I$ and the target $T$ are constant over time. In our simulation however, we have a set of patterns $P_\alpha$ presented to the network. A pattern is a tuple in $([0,1] \cup \{U\})^N$, where $N$ is the total number of units and $U$ stands for unclamped. The $i^{th}$ value of the tuple is the value assigned to $I_i$ and $T_i$ when the pattern is presented to the network (if the value is $U$, the unit is unclamped for the time of presentation of the pattern). This definition of a pattern does not allow $I_{i\alpha}$ and $T_{i\alpha}$ to have different values. This is not an important restriction however since we can can always simulate such an (inconsistent) unit with two units. The energy function to be minimized over all the patterns is defined by the equation:

$$E_{total} = \sum_\alpha E(\alpha) \tag{9}$$

The gradient of $E_{total}$ is simply the sum of the gradients of $E(\alpha)$, and hence the updating equation has the form:

$$\mathrm{d}w_{ij}/\mathrm{d}t = \eta \sum_\alpha y_i^\infty(\alpha)x_j^\infty(\alpha) \tag{10}$$

When a pattern $P_\alpha$ is presented to the network, an approximation of $x_j^\infty(\alpha)$ is first computed by doing a few iterations using equation 1 (propagation). Then, an approximation of $y^\infty(\alpha)$ is evaluated by iterating equation 8 (backpropagation). The weights are finally updated using equation 10. If we assume the number of iterations to evaluate $x_j^\infty(\alpha)$ and $y_j^\infty(\alpha)$ to be constant, the total number of operations required to update the weights is $O(N^2)$. The validity of this assumption will be discussed in a later section.

## CONVERGENCE OF THE NETWORK

The learning algorithm for our network assumes a correct approximation of $x^\infty$. This value is computed by recursively propagating the activation signals according to equation 1. The effect of varying the number of propagations can be illustrated with a simple experiment. Consider a fully connected network of eight units (it's a directed anti-reflexive graph). Four of them are auto-associative units which are presented various patterns of zeroes and ones. An auto-associative unit is best viewed as two visible units, one having all of the incoming connections and one having all of the outgoing connections. When the auto-associative unit is not clamped, it is viewed as a hidden unit. The four remaining units are hidden. The error is measured by the differences between the activations (from the incoming connections) of the auto-associative units and the corresponding target value $T_i$ for each pattern. In running the experiment, eight patterns were presented to the network perfo[213zrming 1 to 5 propagations of the activations using Equation 1, 20 back-propagations of the error signals according to Equation 8, and one update (Equation 10) of the weights per presentation. We define an epoch to be a sweep through the eight patterns using the above formula of execution on each. The corresponding results using a learning rate of 0.2 are shown in figure 1. It can easily be seen that using one or two propagations does not suffice to set the hidden units to their correct values. However, the network does learn correctly how to reproduce the eight patterns when 3 or more

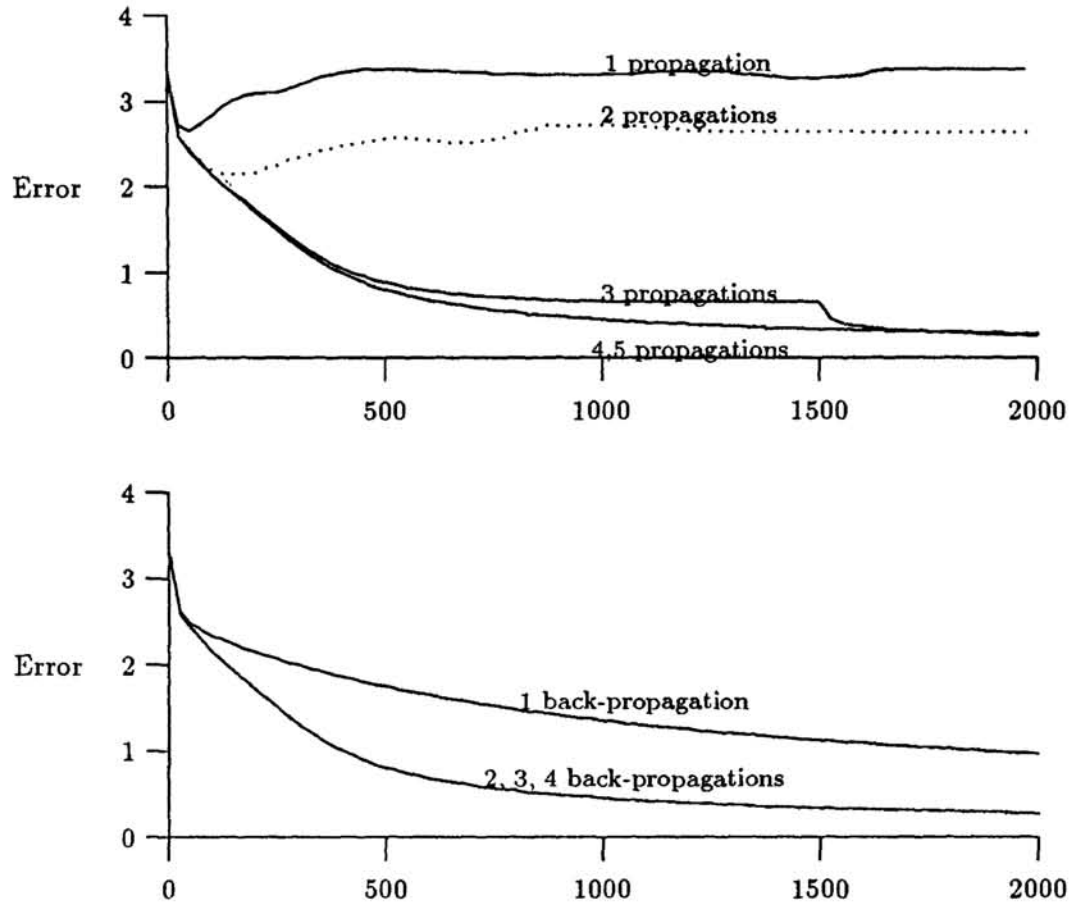

Figure 1: Learning curves for a recurrent network with different numbers of propagations of the activation and back-propagation of the error signals.

propagations are done after each presentation of a new pattern. This is not surprising since the rate of convergence to a fixed point is geometric (if the fixed point is stable), thus making only a few propagations necessary. We suspect that larger networks with a fully connected topology will still only require a few iterations of forward propagation if the fixed points are fairly stable. In the next section, we will study a problem, where this assumption is not true. In such a situation, we use an algorithm where the number of forward propagations varies dynamically.

For some specialized networks such as a feed-forward one, the number of propagations must be at least equal to the number of layers, so that the output units receive the activation corresponding to the input before the error signal is sent.

Similarly, $y^{\infty}$ is computed recursively by iterative steps. We used the same experiment as described above with 1 to 4 back-propagations of the error signals to evaluate the time $y^t$ takes to converge. The rest of this experiment remained the same as above, except that the number of propagations for $x^t$ was set to 20. The learning curves are shown in figure 1. It is interesting to note that with only one propagation of the error signal, the system was able to complete the learning, for the isolated curve tends toward the other curves as time increases. The remaining four curves lie along the same path because the error signals rapidly become meaningless after few iterations. The reason for this is that the error signals are multiplied by $g_i'(u_i) = w_{ij}x_i(1 - x_i)$ when going from unit $i$ to unit $j$, which is usually much smaller than one because $\mid x_i(1 - x_i) \mid$ is smaller than 0.25. The fact that one iteration of the error signal is enough to provide learning is interesting for VLSI applications: it enables the units to work together in an asynchronous fashion. If each unit propagates the activation much more often than it backpropagates the error signals the system is, on average, in a stable state when the backpropagation occurs and the patterns are learned slowly. This ability for recurrent networks to work without synchronization mechanisms makes them more compatible with physiological network systems.

The above discussion assumed that $x^{\infty}$ exists and can be computed by recursively computing the activation function. However, it has been shown ([Simard 88]) that for any activation function, there are always sets of weights such that there exist no stable fixed points. This fact is alarming since $x^{\infty}$ is computed recursively, which implies that if there is no stable fixed point, $x^t$ will fail to converge, and incorrect error signals will be propagated through the system. Fortunately, the absence of stable fixed points turns out not to be a problem in practice. One reason for this is that they are very likely to be present given a reasonable set of initial weights. The network almost always starts with a unique stable fixed point. The fixed points are searched by following the zero curve of the homotopy map

$$\lambda([x_i] - F([x_i])) + (1 - \lambda)([x_i] - [a_i]) \tag{11}$$

for different $[a_i]$ starting at $\lambda = 0$. The results indicate that the probability of getting unstable fixed points increases with the size of the network. We always found a stable fixed point for networks with less than 50 units. Out of 500 trials of 100 unit networks starting with random weights between -1 and +1, we found two set of weights with no stable fixed points. However, even in that case, most of the eigenvalues were much less than 1, which means that oscillations are limited to one or two eigenvector axes.

Since it is possible to start with a network that has no stable fixed points, it is of interest whether it will still learn correctly. Since searching for all the fixed points (by trying different $[a_i]$ in equation 11) is computationally expensive, we choose, as before, a simple learning experiment. The network's layout is the same as previously described. However, we know (from the previous result) than it probably has no unstable fixed points

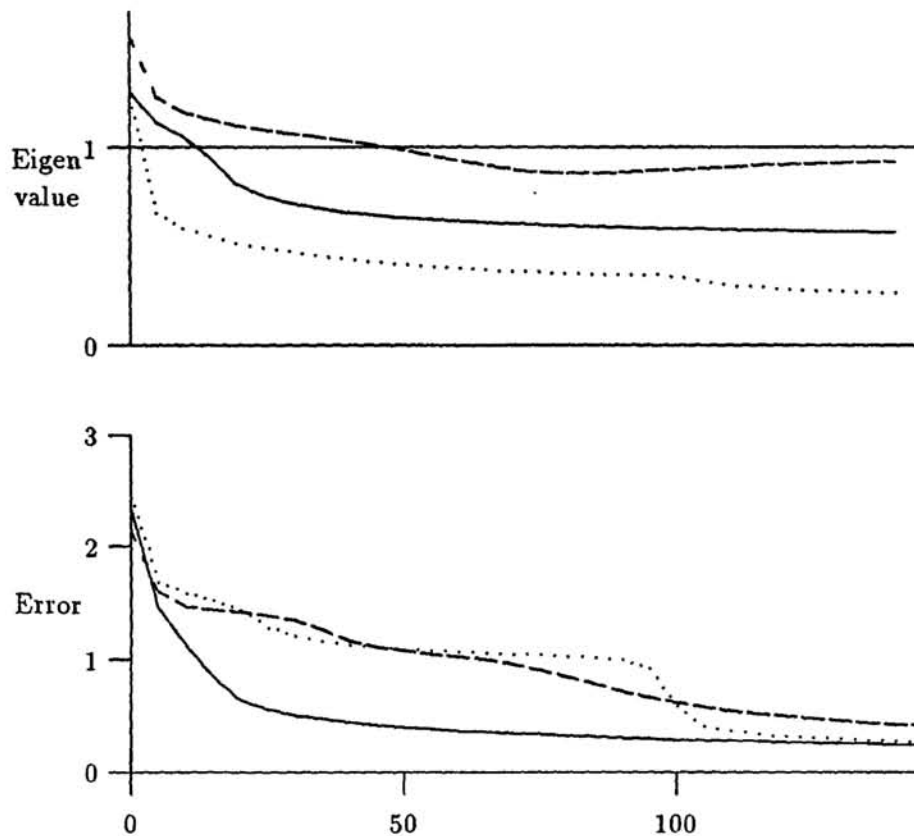

Figure 2: *top* Maximum eigenvalues for the unstable fixed point as a function of the number of epochs. *bottom* Error as a function of the number of epochs.

since it only has four hidden units. To increase the probability of getting a fixed point that is unstable, we make the initial weights range from -3 to 3 and set the thresholds so that [0.5] is a fixed point for one of the patterns. This fixed point is more likely to be unstable since the partial derivative of the functions (which are equal to $\partial g_i(u_i)/\partial x_j = w_{ij}x_i(1-x_i)$ at the fixed point) are maximized at $[x_i] = [0.5]$ and therefore the Jacobian is more likely to have big eigenvalues. Figure 2 shows the stability of that particular fixed point and the error as a function of the number of epochs. Three different simulations were done with different sets of random initial weights. As clearly shown in the figure, the network learns despite the absence of stable fixed points. Moreover, the observed fixed point(s) become stable as learning progresses. In the absence of stable fixed points, the weights are modified after a fixed number of propagations and backpropagations. Even though the state vector of the network is not precisely defined, the state space trajectory lies in a delimited volume. As learning progresses, the projection of this volume on the visible units diminishes to a single point (stable) and moves toward a target point that correspond to the presented pattern on the visible units. Note that our energy function does not impose constraints on the state space trajectories projected on the hidden units [Pearlmutter 88].

## "RUNAWAY" SIMULATIONS

The next question that arises is whether a recurrent network goes to the same fixed point at successive epochs (for a given input) and what happens if it does not. To answer this question, we construct two networks, one with only feed forward connections and one with feed back connections. Both networks have 3 modules (input, hidden and output) of 4 units each. The connections of the feed forward network are between the input and the hidden module and between the hidden and the output module. The connections of the recurrent network are identical except that the there are connections between the units of the hidden module. The rationale behind this layout is to ensure fairness of comparison between feed forward and feedback backpropagation. Each network is presented sixteen distinct patterns on the input with sixteen different random patterns on the output. The patterns consist of zeros and ones. This task is purposely chosen to be fairly difficult (16 fixed points on the four hidden units for the recurrent net) and will make the evaluation of $x^\infty$ difficult. The learning curves for the networks are shown in Figure 3 for a learning rate of 0.2. We can see that the network with recurrent connections learn a slightly faster than the feed forward network. However, a more careful analysis reveals that when the learning rate is increased, the recurrent network doesn't always learn properly. The success of the learning depends on the number of iterations we use in the computation of $x^t$. As clearly shown on the Figure 3, if we use 30 iterations for $x^t$ the network fails to learn, although 40 iterations yields reasonable results. The two cases only differ by the value of $x^t$ used when the error signals are backpropagated.

According to our interpretation, recurrent backpropagation learns by moving the fixed points (or small volume state trajectories) toward target values (determined by the output). As learning progresses, the distances between the fixed points and the target values diminish, causing the error signals to become smaller and the learning to slow down. However if the network doesn't come close enough to the fixed point (or the small volume state trajectory), the new error (the distance between the current state and the target) can suddenly be very large (relatively to the distance between the fixed point and the target). Large incorrect error signals are then introduced into the system. There are two cases: if the learning rate is small, a near miss has little effect on the learning curve and RBP learns faster than the feed forward network. If, on the other hand, the learning rate

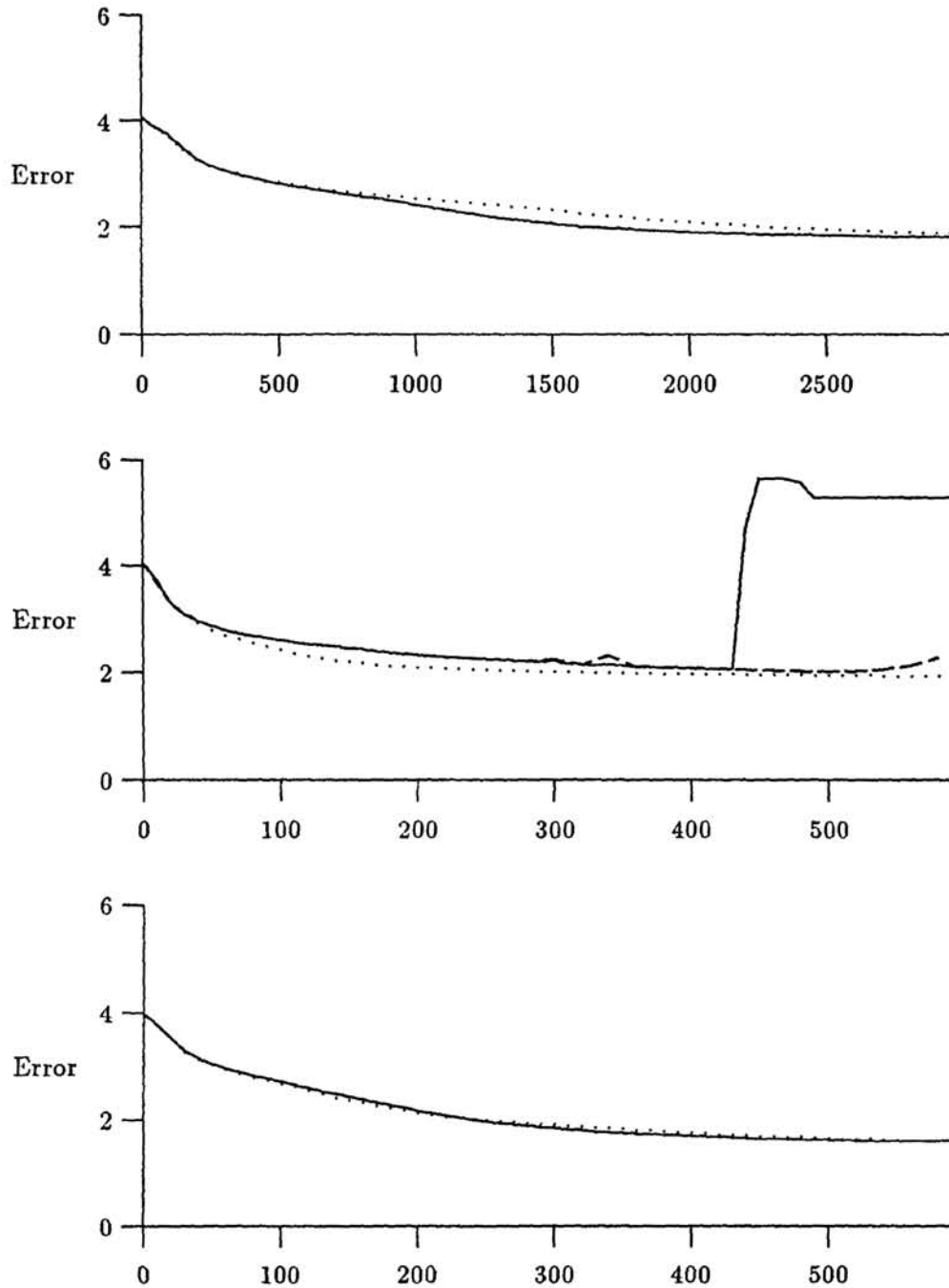

Figure 3: Error as a function of the number of epochs for a feed forward net (dotted) and a recurrent net (solid or dashed). *top:* The learning rate is set to 0.2. *center:* The learning rate is set to 1.0. The solid and the dashed lines are for recurrent net with 30 and 40 iterations of $x^t$ per epochs respectively. *bottom:* The learning rate is variable. The recurrent network has a variable number of iteration of $x^t$ per epochs.

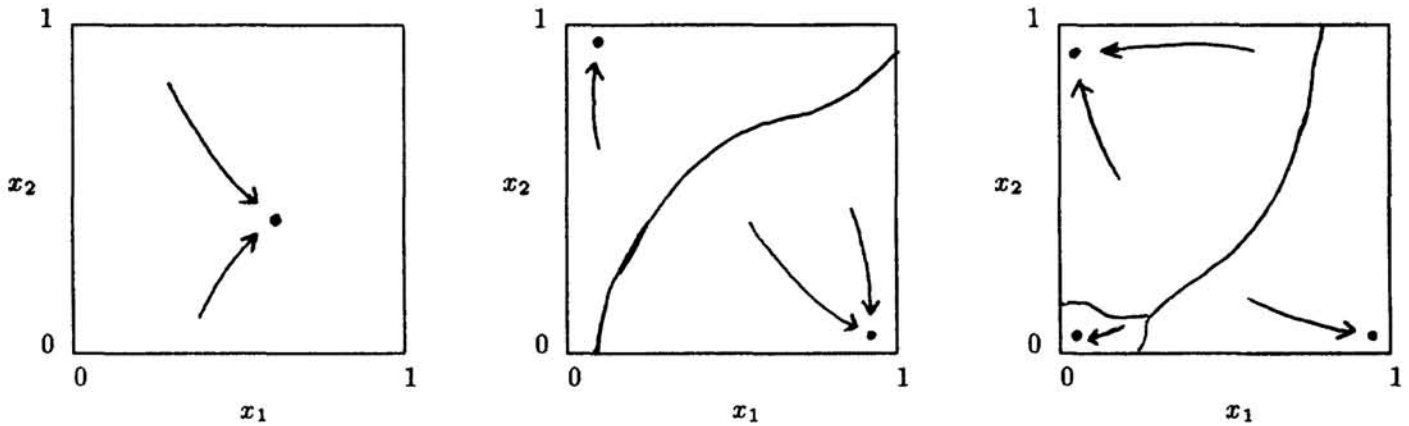

Figure 4: State space and fixed point. $x_1$ and $x_2$ are the activation of two units of a fully connected network. *left:* Before learning, there is one stable fixed point *center:* After learning a few pattern, there are two desired stable fixed points. *right:* After learning several patterns, there are two desired stable fixed points and one undesired stable fixed point.

is big, a near miss will induce important incorrect error signals into the system which in turn makes the next miss more dramatic. This runaway situation is depicted on the center of Figure 3. To circumvent this problem we vary the number of propagations as needed until successive states on the state trajectory are sufficiently close. The resulting learning curves for feed forward and recurrent nets are plotted at the bottom of Figure 3. In these simulations the learning rates are adjusted dynamically so that successive error vectors are almost colinear, that is:

$$0.7 < \cos(\widehat{\Delta w_{ij}^{t}, \Delta w_{ij}^{t+1}}) < 0.9 \tag{12}$$

As can be seen recurrent and feed forward nets learn at the same speed. It is interesting to mention that the average learning rate for the recurrent net is significantly smaller ($\approx 0.65$) than for the feed forward net ($\approx 0.80$). Surprisingly, this doesn't affect the learning speed.

## CONTENT ADDRESSABLE MEMORIES

An interesting property of recurrent networks is their ability to generate fixed points that can be used to perform content addressable memory [Lapedes 86, Pineda 87]. Initially, a fully connected network usually has only one stable fixed point (all units unclamped) (see Figure 4, left). By clamping a few (autoassociative) units to given patterns, it is possible, by learning, to create stable fixed points for the unclamped network (Figure 4, center). To illustrate this property, we build a network of 6 units: 3 autoassociative units

| fixed points | | | | | | Maximum |
| autoassociative units | | | hidden units | | | eigenvalue |
| --- | --- | --- | --- | --- | --- | --- |
| 0.0402 | 0.0395 | 0.9800 | 0.8699 | 0.0763 | 0.0478 | 0.4419 |
| 0.9649 | 0.0476 | 0.0450 | 0.0724 | 0.8803 | 0.4596 | 0.6939 |
| 0.0830 | 0.9662 | 0.0658 | 0.2136 | 0.0880 | 0.8832 | 0.8470 |
| 0.9400 | 0.9619 | 0.9252 | 0.1142 | 0.1692 | 0.5164 | 0.8941 |
| 0.9076 | 0.5201 | 0.0391 | 0.0448 | 0.6909 | 0.7431 | 1.2702 |

Table 1: Fixed points for content addressable memory

and 3 hidden units. The three autoassociative units are presented patterns with an odd number of ones in them (there are 4 such patterns on 3 units: 1 0 0, 0 1 0, 0 0 1 and 1 1 1). The network is fully connected. After 5000 epochs, the auto-associative units are unclamped for testing. All the fixed points found for the network of 6 (unclamped) units are given in table 1. As can be seen, the four stable fixed points are exactly the four patterns presented to the network. Moreover their stability guarantees that the network can be used for CAM (content addressable memory) or for one-to-many function learning. Indeed, if the network is presented incomplete or corrupted patterns (sufficiently close to a previously learned pattern), it will restore the pattern as soon as the incorrect or missing units are unclamped by converging to a stable fixed point. If there are several correct pattern completions for the clamped units, the network will converge to one of the pattern depending on the initial conditions of the unclamped units (which determine the state space trajectory). These highly desirable properties are the main advantages of having feedback connections. We note from table 1 that a fifth (incorrect) fixed point has also be found. However, this fixed point is unstable (Maximum eigenvalue = 1.27) and will therefore never be found during recursive searches.

In the previous example, there are no undesired stable fixed points. They are, however, likely to appear if the learning task becomes more complex (Figure 4, right). The reason why they are difficult to avoid is that unless the units are unclamped (the learning is stopped), the network cannot reach them. Algorithms which eliminate spurious fixed points are presently under study.

## CONCLUSION

In this paper, we have studied the effect of introducing feedback connections into feed forward networks. We have shown that the potential disadvantages of the algorithm, such as the absence of stable fixed points and chaotic behavior, can be overcome. The resulting systems have several interesting properties. First, allowing arbitrary connections makes a network more physiologically plausible by removing structural constraints on the topology. Second, the increased number of connections diminishes the sensitivity to noise and slightly improves the speed of learning. Finally, feedback connections allow the network to restore incomplete or corrupted patterns by following the state space trajectory to a stable fixed point. This property can also be used for one-to-many function learning. A limitation of the algorithm, however, is that spurious stable fixed points could lead to incorrect pattern completion.

# References

[Almeida 87]  Luis B. Almeida, in the Proceedings of the IEEE First Annual International Conference on Neural Networks, San Diego, California, June 1987.

[Lapedes 86]  Alan S. Lapedes & Robert M. Farber A self-optimizing nonsymmetrical neural net for content addressable memory and pattern recognition. Physica D22, 247-259, 1986.

[Lippman 87]  Richard P. Lippman, An introduction to computing with neural networks, IEEE ASSP Magazine April 1987.

[Jordan 88]  Michael I. Jordan, Supervised learning and systems with excess degrees of freedom. COINS Technical Report 88-27. Massachusetts Institute of Technology. 1988.

[Pearlmutter 88]  Barak A. Pearlmutter. Learning State Space Trajectories in Recurrent Neural Networks. Proceedings of the Connectionnist Models Summer School. pp. 113-117. 1988.

[Pineda 87]  Fernando J. Pineda. Generalization of backpropagation to recurrent and higher order neural networks. Neural Information Processing Systems, New York, 1987.

[Pineda 88]  Fernando J. Pineda. Dynamics and Architecture in Neural Computation. Journal of Complexity, special issue on Neural Network. September 1988.

[Simard 88]  Patrice Y. Simard, Mary B. Ottaway and Dana H. Ballard, Analysis of recurrent backpropagation. Technical Report 253. Computer Science, University of Rochester, 1988.

[Rosenblatt 62]  F. Rosenblatt, Principles of Neurodynamics, New York: Spartam Books, 1962.

[Rumelhart 86]  D. E. Rumelhart, G. E. Hinton, & R. J. Williams, Learning internal representations by back-propagating errors. Nature, 323,533-536.